# The Role of Lateral Cortical Competition in Ocular Dominance Development

**Christian Piepenbrock and Klaus Obermayer**
Dept. of Computer Science, Technical University of Berlin
FR 2-1; Franklinstr. 28–29; 10587 Berlin, Germany '
{piep,oby}@cs.tu-berlin.de; http://www.ni.cs.tu-berlin.de

## Abstract

Lateral competition within a layer of neurons sharpens and localizes the response to an input stimulus. Here, we investigate a model for the activity dependent development of ocular dominance maps which allows to vary the degree of lateral competition. For weak competition, it resembles a correlation-based learning model and for strong competition, it becomes a self-organizing map. Thus, in the regime of weak competition the receptive fields are shaped by the second order statistics of the input patterns, whereas in the regime of strong competition, the higher moments and "features" of the individual patterns become important. When correlated localized stimuli from two eyes drive the cortical development we find $(i)$ that a topographic map and binocular, localized receptive fields emerge when the degree of competition exceeds a critical value and $(ii)$ that receptive fields exhibit eye dominance beyond a second critical value. For anti-correlated activity between the eyes, the second order statistics drive the system to develop ocular dominance even for weak competition, but no topography emerges. Topography is established only beyond a critical degree of competition.

## 1  Introduction

Several models have been proposed in the past to explain the activity depending development of ocular dominance (OD) in the visual cortex. Some models make the ansatz of linear interactions between cortical model neurons [2, 7], other approaches assume competitive winner-take-all dynamics with intracortical interactions [3, 5]. The mechanisms that lead to ocular dominance critically depend on this choice. In linear activity models, second order correlations of the input patterns determine the receptive fields. Nonlinear competitive models like the self-organizing map, however, use higher order statistics of the input stimuli and map their features. In this contribution, we introduce a general nonlinear

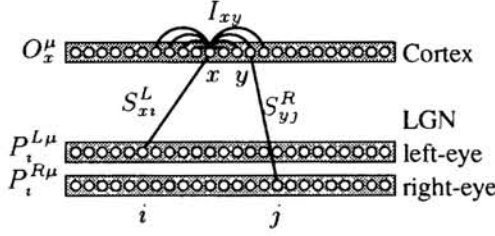

Figure 1: Model for OD development: the input patterns $P_i^{L\mu}$ and $P_i^{R\mu}$ in the LGN drive the Hebbian modification of the cortical afferent synaptic weights $S_{xi}^L$ and $S_{xi}^R$. Cortical neurons are in competition and interact with effective strengths $I_{xy}$. Locations in the LGN are indexed $i$ or $j$, cortical locations are labeled $x$ or $y$.

Hebbian development rule which interpolates the degree of lateral competition and allows us to systematically study the role of non-linearity in the lateral interactions on pattern formation and the transition between two classes of models.

## 2  Ocular Dominance Map Development by Hebbian Learning

Figure 1 shows our basic model framework for ocular dominance development. We consider two input layers in the lateral geniculate nucleus (LGN). The input patterns $\mu = 1, \ldots, U$ on these layers originate from the two eyes and completely characterize the input statistics (the mean activity $\bar{P}$ is identical for all input neurons). The afferent synaptic connection strengths of cortical cells develop according to a generalized Hebbian learning rule with learning rate $\eta$.

$$\Delta S_{xi}^{L\mu} = \eta \sum_y I_{xy} \bar{O}_y^\mu P_i^{L\mu} \quad \text{subject to} \quad \sum_i (S_{xi}^L)^\nu + (S_{xi}^R)^\nu = \text{const.} \quad \forall x \quad (1)$$

An analogous rule is used for the connections from the right eyes $S_{xi}^R$. We use $\nu = 2$ in the following and rescale the length of each neurons receptive field weight vector to a constant length after a learning step. The model includes effective cortical interactions $I_{xy}$ for the development of smooth cortical maps that spread the output activities $\bar{O}_x^\mu$ in the neighborhood of neuron $x$ (with a mean $\bar{I} = \frac{1}{N} \sum_x I_{xy}$ for $N$ output neurons). The cortical output signals are connectionist neurons with a nonlinear activation function $g(.)$,

$$\bar{O}_y^\mu = g(H_y^\mu) = \frac{\exp(\beta H_y^\mu)}{\sum_z \exp(\beta H_z^\mu)} \quad \text{with} \quad H_y^\mu = \sum_j (S_{yj}^L P_j^{L\mu} + S_{yj}^R P_j^{R\mu}), \quad (2)$$

which models the effect of cortical response sharpening and competition for an input stimulus. The degree of competition is determined by the parameter $\beta$. Such dynamics may result as an effect of local excitation and long range inhibition within the cortical layer [6, 1], and in the limits of weak and strong competition, we recover two known types of developmental models—the correlation based learning model and the self-organizing map.

### 2.1  From Linear Neurons to Winner-take-all Networks

In the limit $\beta \to 0$ of weak cortical competition, the output $\bar{O}_y^\mu$ becomes a linear function of the input. A Taylor series expansion around $\beta = 0$ yields a correlation-based-learning (CBL) rule in the average over all patterns

$$\Delta S_{ix}^L = \eta \beta \sum_{z,j} \frac{1}{N} (I_{xz} - \bar{I})(S_{zj}^L C_{ji}^{LL} + S_{zj}^R C_{ji}^{RL}) + \text{const.,}$$

where $C_{ji}^{RL} = \frac{1}{U} \sum_\mu P_j^{R\mu} P_i^{L\mu}$ is the correlation function of the input patterns. Ocular dominance development under this rule requires correlated activity between inputs from

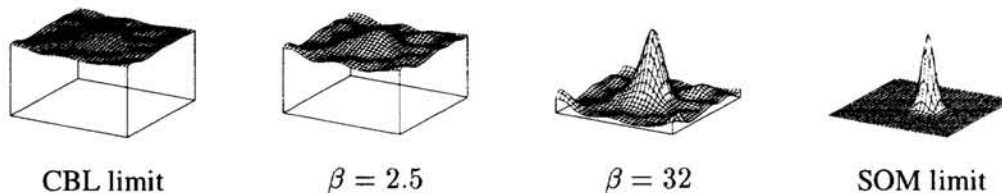

CBL limit          $\beta = 2.5$          $\beta = 32$          SOM limit

Figure 2: The network response for different degrees of cortical competition: the plots show the activity rates $\sum_y I_{xy} \tilde{O}_y^\mu$ for a network of cortical output neurons (the plots are scaled to have equal maxima). Each gridpoint represents the activity of one neuron on a $16 \times 16$ grid. The interactions $I_{xy}$ are Gaussian (variance 2.25 grid points) and all neurons are stimulated with the same Gaussian stimulus (variance 2.25). The neurons have Gaussian receptive fields (variance $\sigma^2 = 4.5$) in a topographic map with additive noise (uniformly distributed with amplitude 10 times the maximum weight value).

within one eye and anti-correlated activity (or uncorrelated activity with synaptic competition) between the two eyes [2, 4]. It is important to note, however, that CBL models cannot explain the emergence of a topographic projection. The topography has to be hard-wired from the outset of the development process which is usually implemented by an "arbor function" that forces all non-topographic synaptic weights to zero.

Strong competition with $\beta \to \infty$, on the other hand, leads to a self-organizing map [3, 5],

$$\Delta S_{ix}^{L\mu} = \eta I_{xq(\mu)} P_i^{L\mu} \quad \text{with} \quad q(\mu) = \text{argmax}_y \sum_j (S_{yj}^L P_j^{L\mu} + S_{yj}^R P_j^{R\mu}) .$$

Models of this type use the higher order statistics of the input patterns and map the important features of the input. In the SOM limit, the output activity pattern is identical in shape for all input stimuli. The input influences only the *location* of the activity on the output layer but does not affect its shape.

For intermediate values of $\beta$, the *shape* of the output activity patterns depends on the input. The activity of neurons with receptive fields that match the input stimulus better than others is amplified, whereas the activity of poorly responding neurons is further suppressed as shown in figure 2. On the one hand, the resulting output activity profiles for intermediate $\beta$ may be biologically more realistic than the winner-take-all limit case. On the other hand, the difference between the linear response case (low $\beta$) and the nonlinear competition (intermediate $\beta$) is important in the Hebbian development process—it yields qualitatively different results as we show in the next section.

## 2.2 Simulations of Ocular Dominance Development

In the following, we study the transition from linear CBL models to winner-take-all SOM networks for intermediate values of $\beta$. We consider input patterns that are localized and show ocular dominance

$$P_i^{L\mu} = \frac{0.5 + eye^L(\mu)}{2\pi\sigma^2} \exp\left(-\frac{(i - loc(\mu))^2}{2\sigma^2}\right) \quad \text{with} \quad eye^L(\mu) = -eye^R(\mu) \quad (3)$$

Each stimulus $\mu$ is of Gaussian shape centered on a random position $loc(\mu)$ within the input layer and the neuron index $i$ is interpreted as a two-dimensional location vector in the input layer. The parameter $eye(\mu)$ sets the eye dominance for each stimulus. $eye = 0$ produces binocular stimuli and $eye = \pm\frac{1}{2}$ results in uncorrelated left and right eye activities.

We have simulated the development of receptive fields and cortical maps according to equations 1 and 2 (see figure 3) for square grids of model neurons with periodic boundary conditions, Gaussian cortical interactions, and OD stimuli (equation 3). The learning

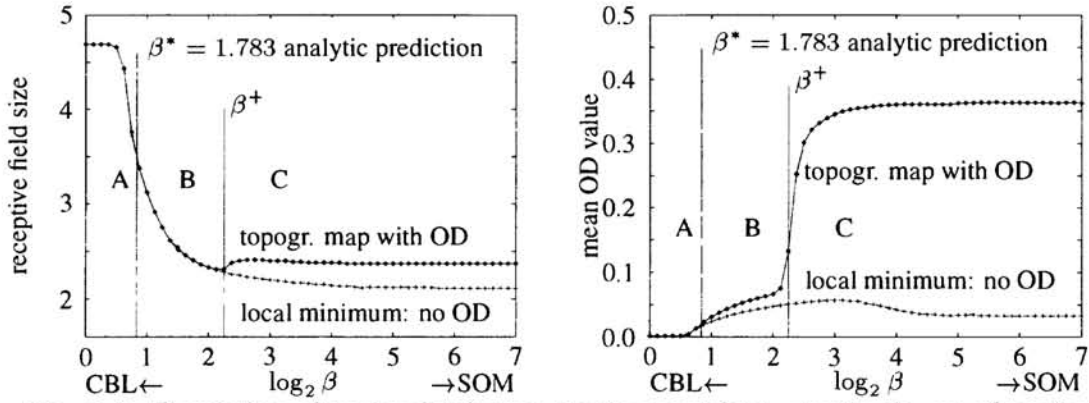

Figure 3: Simulation of ocular dominance development for a varying degree of cortical competition $\beta$ in a network of $16 \times 16$ neurons in each layer. The figure shows receptive fields sizes (left) and mean OD value (right) as a function of cortical competition $\beta$. Each point in the figure represents one simulation with 30000 pattern presentations. The cortical interactions are Gaussian with a variance of $\gamma^2 = 2.25$ grid points. The Gaussian input stimuli are 5.66 times stronger in one eye than in the other (equation 3 with $\sigma^2 = 2.25$, $eye(\mu) = \pm 0.35$). The synaptic weights are intialized with a noisy topographic map (curves labeled "no OD") and additionally with ocular dominance stripes (curves labeled "with OD"). To determine the receptive field size we have applied a Gaussian fit to all receptive field profiles $S_{ix}^L$ and $S_{ix}^L$ and averaged the standard deviation (in grid points) over all neurons $x$. The mean OD value is given by $\frac{1}{N} \sum_x |\sum_i (S_{ix}^L - S_{ix}^R) / \sum_i (S_{ix}^L + S_{ix}^R)|$.

rate is set at the first stimulus presentation to change the weights of the best responding neuron by half a percent. After each learning step the weights are rescaled to enforce the constraint from equation 1.

The simulations yield the results expected in the CBL and SOM limit cases (small and large $\beta$) for initially constant synaptic weight values with 5 percent additional noise. In the CBL limit, our particular choice of input patterns does not lead to the development of ocular dominance, because the necessary conditions for the input pattern correlations are not satisfied—the pattern correlations and interactions are all positive. Instead, the learning rule has only one fixpoint with uniform synaptic weights—unstructured receptive fields that cover the whole input layer. In the SOM limit, our set of stimuli leads to the emergence of a topographic projection with localized receptive fields and ocular dominance stripes. The topographic maps often develop defects which can be avoided by an annealing scheme. Instead of annealing $\beta$ or the cortical interaction range, however, we initialize the weights with a topographic projection and some additive noise. This is a common assumption in cortical development models [2], because the fibers from the LGN first innervate the visual cortex already in a coarsely topographic order.

For intermediate degrees of cortical competition, we find sharp transitions between the CBL and SOM states and distinguish three parameter regimes (see figure 3). For weak competition (A) all receptive fields are unstructured and cover the whole input layer. At some critical $\beta^*$, the receptive fields begin to form a topographic projection from the geniculate to the cortical layer. This projection (B) has no stable ocular dominance stripes, but a small degree of ocular dominance that fluctuates continuously. For yet stronger competition (C), a cortical map with stable ocular dominance stripes emerges.

The simulations, however, show that a topographic map without ocular dominance remains a stable attractor of the learning dynamics (C). For increasing competition its basin of attraction becomes smaller, and smaller learning rates are necessary in order to remain within the binocular state. On the one hand, simulations with slowly *increasing beta* lead to a to-

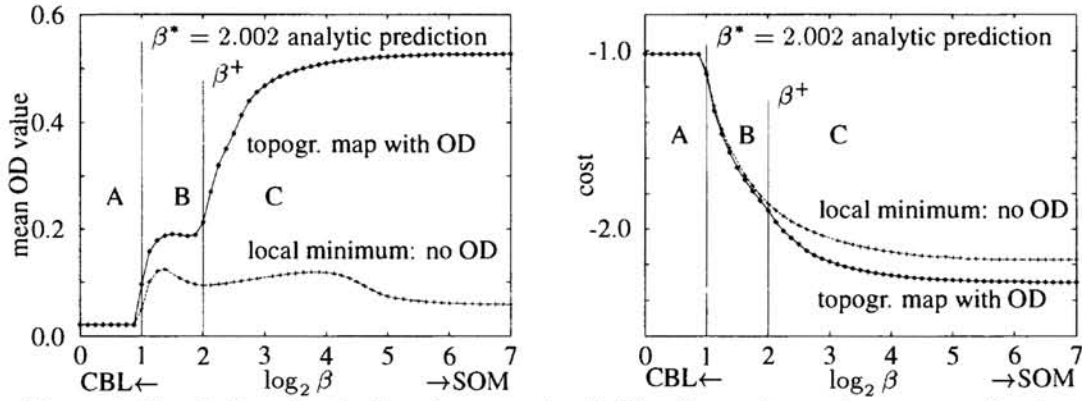

Figure 4: Simulations for the learning equation 5. The figure shows the mean ocular dominance (left) and the cost (right) as a function of $\beta$. The parameters are identical to figure 3 and $eye(\mu) = \pm 0.425$.

pographic map and ocular dominance stripes suddenly pop up somewhere in regime C—for small learning rates later than for large ones. On the other hand, in simulations with *decreasing* $\beta$ and an initially topographic map with ocular dominance, we find a second critical $\beta^+$ at which the OD map becomes unstable.

To understand the system's properties better, we analytically predict the value $\beta^*$—the point where structured receptive fields emerge—and discuss the relation to cost functions to get some intuition about the value $\beta^+$ in the following paragraph.

### 2.3  Analysis of the Emergence of Structured Receptive Fields

For $\beta < \beta^*$ the system shows basically CBL properties—in our case constant weights and unstructured receptive fields. It is possible to study the stability of this state analytically. We consider the learning equation 1 under a hard renormalization constraint that enforces $\sum_{i=1}^{M}(S_{xi}^L)^2 + (S_{xi}^R)^2 = 2M\bar{S}^2$ by rescaling the weights after each learning step. A linear perturbation analysis of the learning rule around constant weights yields a critical degree of competition $\beta^* = (\bar{S}\lambda_{\max}^K\lambda_{\max}^I)^{-1}$ where $\bar{S}$ is the strength of the constant synaptic weights. $\lambda_{\max}^K$ is the largest eigenvalue of the input covariance matrix

$$\frac{1}{\bar{P}}\left(\begin{bmatrix} C_{ji}^{LL} & C_{ji}^{LR} \\ C_{ji}^{RL} & C_{ji}^{RR} \end{bmatrix} - \bar{P}^2\right),$$

which has to be diagonalized with respect to $L$ and $R$, as well as with respect to $i$ and $j$. The input correlation functions for the patterns from equation 3 are given by $C_{ji}^{LL} = C_{ji}^{RR} = (\frac{1}{2} + 2eye^2)\frac{1}{U}G(i-j, 2\sigma^2)$ and $C_{ji}^{LR} = C_{ji}^{RL} = (\frac{1}{2} - 2eye^2)\frac{1}{U}G(i-j, 2\sigma^2)$ where $G(\vec{i}, \sigma^2)$ is a two-dimensional Gaussian with variance $\sigma^2$. The eigenvalues with respect to $L$ and $R$ in this symmetric case are the sum and difference terms of the correlation functions $K_{ij}^{\text{sum}} = \frac{1}{\bar{P}}(C_{ji}^{LL} + C_{ji}^{LR} - \bar{P}^2)$ and $K_{ij}^{\text{diff}} = \frac{1}{\bar{P}}(C_{ji}^{LL} - C_{ji}^{LR})$. The term $K_{ij}^{\text{sum}}$ is larger for positive input correlations and in the next step we have to find the eigenvalues of this matrix. For periodic boundary conditions and in the limit of large networks, we can approximate the eigenvalue by the fourier transform of the Gaussian and finally obtain $\lambda_{\max}^K = \exp\left(-(\sigma 2\pi/m)^2\right)$ (for a square grid of $M = m \times m$ neurons). $\lambda_{\max}^I$ is the largest eigenvalue of $(\frac{1}{NI}I_{xz} - \frac{1}{N})$ and Gaussian cortical interactions $I_{xy}$ with variance $\gamma^2$ on $N = n \times n$ output neurons yield $\lambda_{\max}^I = \exp\left(-\frac{1}{2}(\gamma 2\pi/n)^2\right)$. Stronger competition beyond the point $\beta^*$ leads to the formation of structured receptive fields. It is interesting to note, that the critical $\beta^*$ does not depend on $eye(\mu)$, the strength of ocularity in the input patterns. The predicted value for $\beta^*$ is plotted in figure 3 and matches the transition found in the simulations.

### 2.4  Hebbian Development With a Global Objective Function

The learning equation 1 does not optimize a global cost function [5]. To understand the dynamics of the OD development better and to interpret the transition at $\beta*$, we derive a learning rule very similar to equation 1 that minimizes the global cost function $E$,

$$E = \frac{1}{U} \sum_{\mu,x} O_x^\mu \text{cost}_x^\mu \quad \text{with} \quad \text{cost}_x^\mu = -\sum_{y,j} I_{xy}(S_{yj}^L P_j^{L\mu} + S_{yj}^R P_j^{R\mu}) \, . \tag{4}$$

We minimize this cost function in a stochastic network of binary output neurons $O_x^\mu$ that compete for the input stimuli, i.e. one output neuron is active at any given time. The probability for a neuron $y$ to become active in response to pattern $\mu$ depends on its advantage in cost over the currently active neuron $x$:

$$P(O_x^\mu = 1 \to O_y^\mu = 1) = \frac{\exp[-\beta(\text{cost}_x^\mu - \text{cost}_y^\mu)]}{\sum_z \exp[-\beta(\text{cost}_x^\mu - \text{cost}_z^\mu)]} \, .$$

This type of output dynamics leads to a Boltzmann probability distribution for the state of the system. We marginalize over all possible network outputs and derive a learning rule by gradient descent on the log likelihood of a particular set of synaptic connections (subject to $\sum_i (S_{xi}^L)^n + (S_{xi}^R)^n = \text{const.}$).

$$\Delta S_{xi}^L = \eta \frac{\partial}{\partial S_{xi}^L} \log \text{Prob}(\{S_{xi}^L, S_{xi}^R\}) = \frac{\partial}{\partial S_{xi}^L} \log \sum_{\{O_x^\mu\}} \frac{1}{Z} \exp(-\beta E) \, .$$

Finally, we obtain a learning rule that contains the expectation values $\bar{O}_x^\mu$ (or mean fields) of the binary outputs,

$$\Delta S_{xi}^{L\mu} = \eta \sum_y I_{xy} \bar{O}_y^\mu P_i^{L\mu} \text{ with } \bar{O}_x^\mu = \frac{\exp(\beta \sum_{y,j} I_{xy}(S_{yj}^L P_j^{L\mu} + S_{yj}^R P_j^{R\mu}))}{\sum_z \exp(\beta \sum_{y,j} I_{zy}(S_{yj}^L P_j^{L\mu} + S_{yj}^R P_j^{R\mu}))} \, . \tag{5}$$

This learning rule is almost identical to equation 1, it only contains an additional cortical interaction inside the output term $\bar{O}_x^\mu$, but it has the advantage of an underlying cost function.

Figure 4 shows the development of ocular dominance according to equation 5 and the associated cost is plotted for each state of the system. The value $\beta^*$ of the first transition is calculated analogously to the previous section and $\lambda_{\max}^I$ becomes the maximum eigenvalue of the matrix $(\frac{1}{N\bar{I}} \sum_y I_{xy} I_{yz} - \bar{I})$ which is $\lambda_{\max}^I = \exp\left(-(\gamma 2\pi/m)^2\right)$. Around $\beta^+$ a topographic map without ocular dominance is a stable state and it remains stable for larger $\beta$. In addition, a different minimum of the cost function equation 4 emerges at $\beta^+$: an ocular dominance map with a lower associated cost. This shows that an ocular dominance map becomes the preferred state of the system beyond $\beta^+$ although the binocular topographic map is still stable. In the SOM limit $\beta \to \infty$ the binocular topographic map becomes unstable and ocular dominance stripes develop.

The value $\beta^+$ marks the first emergence of an ocular dominance map. For the simulations in the figures 3 and 4 we have used positive correlations between the two eyes—a realistic assumption for OD map development. For weaker correlations ($eye(\mu)$ approaches $\pm\frac{1}{2}$), $\beta^+$ decreases. For anti-correlated stimuli, an ocular dominance map develops even in the CBL limit [4] (this, however, requires additional model assumptions like inhibition between the layers within the LGN). Such a map has no topographic structure (if not imposed by an arbor function) but mostly monocular receptive fields. The value $\beta^*$ is not affected directly by those changes and the monocular receptive fields localize, if $\beta^*$ is exceeded. Consequently, the "feature" OD emerges, if it is dominant in the relevant pattern statistics—for anti-correlated eyes around $\beta = 0$, and for positive between-eye-correlations only in the regime of higher order moments at $\beta^+$.

## 3   Conclusions

We have introduced a model for cortical development with a variable degree of cortical competition. For weak competition it has CBL models, for strong competition the SOM as limit cases. Localized stimuli with ocular dominance require a minimum degree of cortical competition to develop a topographic map, and a stronger degree of competition for the emergence of ocular dominance stripes. Anti-correlated activity between the two eyes lets OD emerge for weak competition and localized fields only beyond a critical degree of competition.

A Taylor series expansion of the learning equation 1 yields a CBL model that uses only second order input statistics. For increasing $\beta$ the higher order terms become significant which consist of the higher moments of the input patterns. In this contribution, we have used only simple activity blobs in two eyes, but it is well known that in the winner-take-all limit features like orientation selectivity can emerge as well [3].

The soft cortical competition in our model implements a mechanism of response sharpening in which the input patterns do still influence the output pattern shape. This should relax the biologically unplausible assumption of winner-take-all dynamics of SOM models and yields similar ocular dominance maps. Cortical microcircuits—local cortical amplifiers— have been proposed as a cortical module of computation [6]. Our model suggests that such circuits may be important to sharpen the responses during the *development* and to permit the emergence of feature mapping simple cell receptive fields.

Our model shows that small changes in the degree of cortical competition may result in qualitative changes of the emerging receptive fields and cortical maps. Such changes in competition could be a result of the maturation of the intra-cortical connectivity. A slowly increasing degree of cortical competition could make the cortical neurons sensitive to more and more complex features of the input stimuli.

### Acknowledgements

This work was supported by the Boehringer Ingelheim Fonds (C. Piepenbrock) and by DFG grant Ob 102/2-1.

### References

[1] S. Amari. Dynamics of pattern formation in lateral-inhibition type neural fields. *Biol. Cyb.*, 27:77–87, 1977.

[2] K. D. Miller, J. B. Keller, and M. P. Stryker. Ocular dominance column development: Analysis and simulation. *Science*, 245:605–615, 1989.

[3] K. Obermayer, H. Ritter, and K. Schulten. A principle for the formation of the spatial structure of cortical feature maps. *Proc. Nat. Acad. Sci. USA*, 87:8345–49, 1990.

[4] C. Piepenbrock, H. Ritter, and K. Obermayer. The joint development of orientation and ocular dominance: Role of constraints. *Neur. Comp.*, 9:959–970, 1997.

[5] M. Riesenhuber, H.-U. Bauer, and T. Geisel. Analyzing phase transitions in high-dimensional self-organizing maps. *Biol. Cyb.*, 75:397–407, 1996.

[6] D. C. Somers, S. B. Nelson, and M. Sur. An emergent model of orientation selectivity in cat visual cortical simple cells. *J. Neurosci.*, 15:5448–5465, 1995.

[7] A. L. Yuille, J. A. Kolodny, and C. W. Lee. Dimension reduction, generalized deformable models and the development of ocularity and orientation. *Neur. Netw.*, 9:309–319, 1996.
